# Rank-Approximate Nearest Neighbor Search: Retaining Meaning and Speed in High Dimensions

**Parikshit Ram, Dongryeol Lee, Hua Ouyang and Alexander G. Gray**
Computational Science and Engineering, Georgia Institute of Technology
Atlanta, GA 30332
`{p.ram@,dongryel@cc.,houyang@,agray@cc.}gatech.edu`

## Abstract

The long-standing problem of efficient nearest-neighbor (NN) search has ubiquitous applications ranging from astrophysics to MP3 fingerprinting to bioinformatics to movie recommendations. As the dimensionality of the dataset increases, exact NN search becomes computationally prohibitive; $(1+\epsilon)$ distance-approximate NN search can provide large speedups but risks losing the meaning of NN search present in the ranks (ordering) of the distances. This paper presents a simple, practical algorithm allowing the user to, for the first time, directly control the true accuracy of NN search (in terms of ranks) while still achieving the large speedups over exact NN. Experiments on high-dimensional datasets show that our algorithm often achieves faster and more accurate results than the best-known distance-approximate method, with much more stable behavior.

## 1 Introduction

In this paper, we address the problem of nearest-neighbor (NN) search in large datasets of high dimensionality. It is used for classification ($k$-NN classifier [1]), categorizing a test point on the basis of the classes in its close neighborhood. Non-parametric density estimation uses NN algorithms when the bandwidth at any point depends on the $k^{th}$ NN distance (NN kernel density estimation [2]). NN algorithms are present in and often the main cost of most non-linear dimensionality reduction techniques (manifold learning [3, 4]) to obtain the neighborhood of every point which is then preserved during the dimension reduction. NN search has extensive applications in databases [5] and computer vision for image search Further applications abound in machine learning.

Tree data structures such as $kd$-trees are used for efficient exact NN search but do not scale better than the naïve linear search in sufficiently high dimensions. Distance-approximate NN (DANN) search, introduced to increase the scalability of NN search, approximates the distance to the NN and any neighbor found within that distance is considered to be "good enough". Numerous techniques exist to achieve this form of approximation and are fairly scalable to higher dimensions under certain assumptions.

Although the DANN search places bounds on the numerical values of the distance to NN, in NN search, distances themselves are not essential; rather the order of the distances of the query to the points in the dataset captures the necessary and sufficient information [6, 7]. For example, consider the two-dimensional dataset $(1, 1), (2, 2), (3, 3), (4, 4), \ldots$ with a query at the origin. Appending non-informative dimensions to each of the reference points produces higher dimensional datasets of the form $(1, 1, 1, 1, 1, \ldots), (2, 2, 1, 1, 1, \ldots), (3, 3, 1, 1, 1, \ldots), (4, 4, 1, 1, 1, \ldots), \ldots$. For a fixed distance approximation, raising the dimension increases the number of points for which the distance to the query (i.e. the origin) satisfies the approximation condition. However, the ordering (and hence the ranks) of those distances remains the same. The proposed framework, *rank-approximate nearest-neighbor* (RANN) search, approximates the NN in its rank rather than in its distance, thereby making the approximation independent of the distance distribution and only dependent on the ordering of the distances.

This paper is organized as follows: Section 2 describes the existing methods for exact NN and DANN search and the challenges they face in high dimensions. Section 3 introduces the proposed approach and provides a practical algorithm using stratified sampling with a tree data structure to obtain a user-specified level of rank approximation in Euclidean NN search. Section 4 reports the experiments comparing RANN with exact search and DANN. Finally, Section 5 concludes with discussion of the road ahead.

## 2    Related Work

The problem of NN search is formalized as the following:

**Problem.** Given a dataset $S \subset X$ of size $N$ in a metric space $(X, d)$ and a query $q \in X$, efficiently find a point $p \in S$ such that

$$d(p, q) = \min_{r \in S} d(r, q). \tag{1}$$

### 2.1    Exact Search

The simplest approach of *linear search* over $S$ to find the NN is easy to implement, but requires $\mathbf{O}(N)$ computations for a single NN query, making it unscalable for moderately large $N$.

Hashing the dataset into buckets is an efficient technique, but scales only to very low dimensional $X$. Hence data structures are used to answer queries efficiently. Binary spatial partitioning trees, like $kd$-trees [9], ball trees [10] and metric trees [11] utilize the triangular inequality of the distance metric $d$ (commonly the Euclidean distance metric) to *prune* away parts of the dataset from the computation and answer queries in expected $\mathbf{O}(\log N)$ computations [9]. Non-binary cover trees [12] answer queries in theoretically bounded $\mathbf{O}(\log N)$ time using the same property under certain mild assumptions on the dataset.

Finding NNs for $\mathbf{O}(N)$ queries would then require at least $\mathbf{O}(N \log N)$ computations using the trees. The dual-tree algorithm [13] for NN search also builds a tree on the queries instead of going through them linearly, hence amortizing the cost of search over the queries. This algorithm shows orders of magnitude improvement in efficiency and is conjectured to be $\mathbf{O}(N)$ for answering $\mathbf{O}(N)$ queries using the cover trees [12].

### 2.2    Nearest Neighbors in High Dimensions

The frontier of research in NN methods is high dimensional problems, stemming from common datasets like images and documents to microarray data. But high dimensional data poses an inherent problem for Euclidean NN search as described in the following theorem:

**Theorem 2.1.** *[8] Let $C$ be a $\mathcal{D}$-dimensional hypersphere with radius $a$. Let $A$ and $B$ be any two points chosen at random in $C$, the distributions of $A$ and $B$ being independent and uniform over the interior of $C$. Let $r$ be the Euclidean distance between $A$ and $B$ ($r \in [0, 2a]$). Then the asymptotic distribution of $r$ is $N(a\sqrt{2}, a^2/2\mathcal{D})$.*

This implies that in high dimensions, the Euclidean distances between uniformly distributed points lie in a small range of continuous values. This hypothesizes that the tree based algorithms perform no better than linear search since these data structures would be unable to employ sufficiently tight bounds in high dimensions. This turns out to be true in practice [14, 15, 16]. This prompted interest in approximation of the NN search problem.

### 2.3    Distance-Approximate Nearest Neighbors

The problem of NN search is relaxed in the following form to make it more scalable:

**Problem.** Given a dataset $S \subset X$ of size $N$ in some metric space $(X, d)$ and a query $q \in X$, efficiently find any point $p' \in S$ such that

$$d(p', q) \leq (1 + \epsilon) \min_{r \in S} d(r, q) \tag{2}$$

for a low value of $\epsilon \in \mathbb{R}^+$ with high probability.

This approximation can be achieved with $kd$-trees, balls trees, and cover trees by modifying the search algorithm to prune more aggressively. This introduces the allowed error while providing some speedup over the exact algorithm [12]. Another approach modifies the tree data structures to

bound error with just one root-to-leaf traversal of the tree, i.e. to eliminate *backtracking*. Sibling nodes in $kd$-trees or ball-trees are modified to share points near their boundaries, forming *spill trees* [14]. These obtain significant speed up over the exact methods. The idea of *approximately correct* (satisfying Eq. 2) NN is further extended to a formulation where the $(1 + \epsilon)$ bound can be exceeded with a low probability $\delta$, thus forming the PAC-NN search algorithms [17]. They provide 1-2 orders of magnitude speedup in moderately large datasets with suitable $\epsilon$ and $\delta$.

These methods are still unable to scale to high dimensions. However, they can be used in combination with the assumption that high dimensional data actually lies on a lower dimensional subspace. There are a number of fast DANN methods that preprocess data with *randomized projections* to reduce dimensionality. *Hybrid spill trees* [14] build *spill trees* on the randomly projected data to obtain significant speedups. *Locality sensitive hashing* [18, 19] hashes the data into a lower dimensional buckets using hash functions which guarantee that "close" points are hashed into the same bucket with high probability and "farther apart" points are hashed into the same bucket with low probability. This method has significant improvements in running times over traditional methods in high dimensional data and is shown to be highly scalable.

However, the DANN methods assume that the distances are well behaved and not concentrated in a small range. However, for example, if the all pairwise distances are within the range (100.0, 101.00), any distance approximation $\epsilon \geq 0.01$ will return an arbitrary point to a NN query. The exact tree-based algorithms failed to be efficient because many datasets encountered in practice suffered the same concentration of pairwise distances. Using DANN in such a situation leads to the loss of the ordering information of the pairwise distances which is essential for NN search [6]. This is too large of a loss in accuracy for increased efficiency. In order to address this issue, we propose a model of approximation for NN search which preserves the information present in the ordering of the distances by controlling the error in the ordering itself irrespective of the dimensionality or the distribution of the pairwise distances in the dataset. We also provide a scalable algorithm to obtain this form of approximation.

## 3  Rank Approximation

To approximate the NN rank, we formulate and relax NN search in the following way:

**Problem.** Given a dataset $S \subset X$ of size $N$ in a metric space $(X, d)$ and a query $q \in X$, let $D = \{D_1, \ldots, D_N\}$ be the set of distances between the query and all the points in the dataset $S$, such that $D_i = d(r_i, q), r_i \in S, i = 1, \ldots, N$. Let $D_{(r)}$ be the $r^{th}$ order statistic of $D$. Then the $r \in S \colon d(r, q) = D_{(1)}$ is the NN of $q$ in $S$. The rank-approximation of NN search would then be to efficiently find a point $p' \in S$ such that

$$d(p', q) \leq D_{(1+\tau)} \tag{3}$$

with high probability for a given value of $\tau \in \mathbb{Z}^+$.

RANN search may use any order statistics of the population $D$, bounded above by the $(1 + \tau)^{th}$ order statistics, to answer a NN query. Sedransk et.al. [20] provide a probability bound for the sample order statistics bound on the order statistics of the whole set.

**Theorem 3.1.** *For a population of size $N$ with $Y$ values ordered as $Y_{(1)} \leq Y_{(2)} \cdots \leq Y_{(N)}$, let $y_{(1)} \leq y_{(2)} \cdots \leq y_{(n)}$ be a ordered sample of size $n$ drawn from the population uniformly without replacement. For $1 \leq t \leq N$ and $1 \leq k \leq n$,*

$$P(y_{(k)} \leq Y_{(t)}) = \sum_{i=0}^{t-k} \left( \begin{array}{c} t-i-1 \\ k-1 \end{array} \right) \left( \begin{array}{c} N-t+i \\ n-k \end{array} \right) / \left( \begin{array}{c} N \\ n \end{array} \right). \tag{4}$$

We may find a $p' \in S$ satisfying Eq. 3 with high probability by sampling enough points $\{d_1, \ldots d_n\}$ from $D$ such that for some $1 \leq k \leq n$, rank error bound $\tau$, and a success probability $\alpha$

$$P(d(p', q) = d_{(k)} \leq D_{(1+\tau)}) \geq \alpha. \tag{5}$$

Sample order statistic $k = 1$ minimizes the required number of samples; hence we substitute the values of $k = 1$ and $t = 1 + \tau$ in Eq. 4 obtaining the following expression which can be computed in $\mathbf{O}(\tau)$ time

$$P(d_{(1)} \leq D_{(1+\tau)}) = \sum_{i=0}^{\tau} \left( \begin{array}{c} N-\tau+i-1 \\ n-1 \end{array} \right) / \left( \begin{array}{c} N \\ n \end{array} \right). \tag{6}$$

The required sample size $n$ for a particular error $\tau$ with success probability $\alpha$ is computed using binary search over the range $(1 + \tau, N]$. This makes RANN search $\mathbf{O}(n)$ (since now we only need to compute the first order statistics of a sample of size $n$) giving $\mathbf{O}(N/n)$ speedup.

## 3.1 Stratified Sampling with a Tree

For a required sample size of $n$, we randomly sample $n$ points from $S$ and compute the RANN for a query $q$ by going through the sampled set linearly. But for a tree built on $S$, parts of the tree would be pruned away for the query $q$ during the tree traversal. Hence we can ignore the random samples from the pruned part of the tree, saving us some more computation.

Hence let $S$ be in the form of a binary tree (say $kd$-tree) rooted at $R_{root}$. The root node has $N$ points. Let the left and right child have $N_l$ and $N_r$ points respectively. For a random query $q \in X$, the population $D$ is the set of distances of $q$ to all the $N$ points in $R_{root}$. The tree stratifies the population $D$ into $D_l = \{D_{l1}, \ldots, D_{lN_l}\}$ and $D_r = \{D_{r1}, \ldots, D_{rN_r}\}$, where $D_l$ and $D_r$ are the set of distances of $q$ to all the $N_l$ and $N_r$ points respectively in the left and right child of the root node $R_{root}$. The following theorem provides a way to decide how much to sample from a particular node, subsequently providing a lower bound on the number of samples required from the unpruned part of the tree without violating Eq.5

**Theorem 3.2.** *Let $n_l$ and $n_r$ be the number of random samples from the strata $D_l$ and $D_r$ respectively by doing a stratified sampling on the population $D$ of size $N = N_l + N_r$. Let $n$ samples be required for Eq.5 to hold in the population $D$ for a given value of $\alpha$. Then Eq.5 holds for $D$ with the same value of $\alpha$ with the random samples of sizes $n_l$ and $n_r$ from the random strata $D_l$ and $D_r$ of $D$ respectively if $n_l + n_r = n$ and $n_l \colon n_r = N_l \colon N_r$.*

*Proof.* Eq. 5 simply requires $n$ uniformly sampled points, i.e. for each distance in $D$ to have probability $n/N$ of inclusion. For $n_l + n_r = n$ and $n_l \colon n_r = N_l \colon N_r$, we have $n_l = \lceil (n/N)N_l \rceil$ and similarly $n_r = \lceil (n/N)N_r \rceil$, and thus samples in both $D_l$ and $D_r$ are included at the proper rate. $\square$

Since the ratio of the sample size to the population size is a constant $\beta = n/N$, Theorem 3.2 is generalizable to any level of the tree.

## 3.2 The Algorithm

The proposed algorithm introduces the intended approximation in the unpruned portion of the $kd$-tree since the pruned part does not add to the computation in the exact tree based algorithms. The algorithm starts at the root of the tree. While searching for the NN of a query $q$ in a tree, most of the computation in the traversal involves computing the distance of the query $q$ to any tree node $R$ ($dist\_to\_node(q, R)$). If the current upperbound to the NN distance ($ub(q)$) for the query $q$ is greater than $dist\_to\_node(q, R)$, the node is traversed and $ub(q)$ is updated. Otherwise node $R$ is pruned. The computations of distance of $q$ to points in the dataset $S$ occurs only when $q$ reaches a leaf node it cannot prune. The NN candidate in that leaf is computed using the linear search (COMPUTEBRUTENN subroutine in Fig.2). The traversal of the exact algorithm in the tree is illustrated in Fig.1.

To approximate the computation by sampling, traversal down the tree is stopped at a node which can be summarized with a small number of samples (below a certain threshold MAXSAMPLES). This is illustrated in Fig.1. The value of MAXSAMPLES giving maximum speedup can be obtained by cross-validation. If a node is summarizable within the desired error bounds (decided by the CANAPPROX-IMATE subroutine in Fig.2), required number of points are sampled from such a node and the nearest neighbor candidate is computed from among them using linear search (COMPUTEAPPROXNN subroutine of Fig.2).

**Single Tree.** The search algorithm is presented in Fig.2. The dataset $S$ is stored as a binary tree rooted at $R_{root}$. The algorithm starts as STRANKAPPROXNN$(q, S, \tau, \alpha)$. During the search, if a leaf node is reached (since the tree is rarely balanced), the exact NN candidate is computed. In case a non-leaf node cannot be approximated, the child node closer to the query is always traversed first. The following theorem proves the correctness of the algorithm.

**Theorem 3.3.** *For a query $q$ and a specified value of $\alpha$ and $\tau$, STRANKAPPROXNN$(q, S, \tau, \alpha)$ computes a neighbor in $S$ within $(1 + \tau)$ rank with probability at least $\alpha$.*

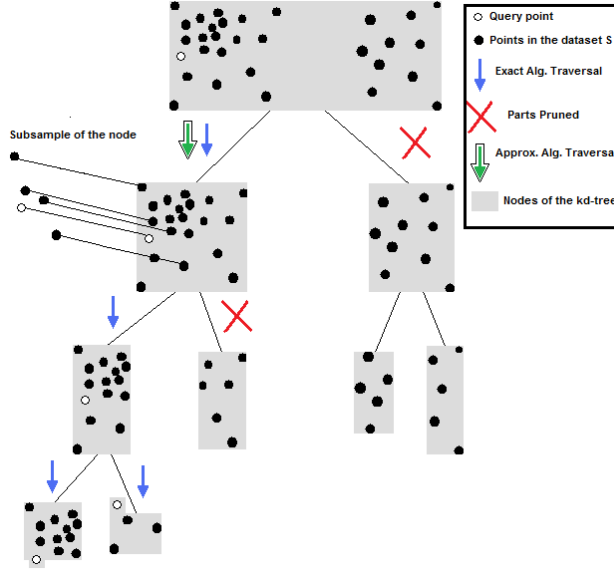

Figure 1: The traversal paths of the exact and the rank-approximate algorithm in a $kd$-tree

*Proof.* By Eq.6, a query requires at least $n$ samples from a dataset of size $N$ to compute a neighbor within $(1 + \tau)$ rank with a probability $\alpha$. Let $\beta = (n/N)$. Let a node $R$ contain $|R|$ points. In the algorithm, sampling occurs when a base case of the recursion is reached. There are three base cases:

- Case 1 - Exact Pruning (**if** $ub(q) \leq dist\_to\_node(q, R)$): Then number of points required to be sampled from the node is at least $\lceil \beta \cdot |R| \rceil$. However, since this node is pruned, we ignore these points. Hence nothing is done in the algorithm.
- Case 2 - Exact Computation COMPUTEBRUTENN$(q, R)$: In this subroutine, linear search is used to find the NN candidate. Hence number of points actually sampled is $|R| \geq \lceil \beta \cdot |R| \rceil$.
- Case 3 - Approximate Computation (COMPUTEAPPROXNN$(q, R, \beta)$): In this subroutine, exactly $\beta \cdot |R|$ samples are made and linear search is performed over them.

Let the total number of points effectively sampled from $S$ be $n'$. From the three base cases of the algorithm, it is confirmed that $n' \geq \lceil \beta \cdot N \rceil = n$. Hence the algorithm computes a NN within $(1+\tau)$ rank with probability at least $\alpha$. □

**Dual Tree.** The single tree algorithm in Fig.2 can be extended to the dual tree algorithm in case of $\mathbf{O}(N)$ queries. The dual tree RANN algorithm (DTRANKAPPROXNN$(T, S, \tau, \alpha)$) is given in Fig.2. The only difference is that for every query $q \in T$, the minimum required amount of sampling is done and the random sampling is done separately for each of the queries. Even though the queries do not share samples from the reference set, when a query node of the query tree prunes a reference node, that reference node is pruned for all the queries in that query node simultaneously. This work-sharing is a key feature of all dual-tree algorithms [13].

## 4 Experiments and Results

A meaningful value for the rank error $\tau$ should be relative to the size of the reference dataset $N$. Hence for the experiments, the $(1 + \tau)$-RANN is modified to $(1 + \lceil \varepsilon \cdot N \rceil)$-RANN where $1.0 \geq \varepsilon \in \mathbb{R}^+$. The Euclidean metric is used in all the experiments. Although the value of MAXSAMPLES for maximum speedup can be obtained by cross-validation, for practical purposes, any low value ($\approx$ 20-30) suffices well, and this is what is used in the experiments.

### 4.1 Comparisons with Exact Search

The speedups of the exact dual-tree NN algorithm and the approximate tree-based algorithm over the linear search algorithm is computed and compared. Different levels of approximations ranging from 0.001% to 10% are used to show how the speedup increases with increase in approximation.

STRANKAPPROXNN($q, S, \tau, \alpha$)

    $n \leftarrow$ COMPUTESAMPLESIZE ($|S|, \tau, \alpha$)
    $\beta \leftarrow n/|S|$
    $R_{root} \leftarrow$ TREE($S$)
    STRANN($q, R_{root}, \beta$)

STRANN($q, R, \beta$)

    **if** $ub(q) > dist\_to\_node(q, R)$ **then**
      **if** ISLEAF($R$) **then**
        COMPUTEBRUTENN($q, R$)
      **else if** CANAPPROXIMATE($R, \beta$)
      **then**
        COMPUTEAPPROXNN ($q, R, \beta$)
      **else**
        STRANN($q, R^l, \beta$),
        STRANN($q, R^r, \beta$)
      **end if**
    **end if**

COMPUTEBRUTENN($q, R$)

    $ub(q) \leftarrow \min(\min_{r \in R} d(q, r), ub(q))$

COMPUTEBRUTENN($Q, R$)

    **for** $\forall q \in Q$ **do**
      $ub(q) \leftarrow \min(\min_{r \in R} d(q, r), ub(q))$
    **end for**
    $node\_ub(Q) \leftarrow \max_{q \in Q} ub(q)$

COMPUTEAPPROXNN($q, R, \beta$)

    $R' \leftarrow \lceil \beta \cdot |R| \rceil$ samples from $R$
    COMPUTEBRUTENN($q, R'$)

COMPUTEAPPROXNN($Q, R, \beta$)

    **for** $\forall q \in Q$ **do**
      $R' \leftarrow \lceil \beta \cdot |R| \rceil$ samples from $R$
      COMPUTEBRUTENN($q, R'$)
    **end for**
    $node\_ub(Q) \leftarrow \max_{q \in Q} ub(q)$

DTRANKAPPROXNN($T, S, \tau, \alpha$)

    $n \leftarrow$ COMPUTESAMPLESIZE ($|S|, \tau, \alpha$)
    $\beta \leftarrow n/|S|$
    $R_{root} \leftarrow$ TREE($S$)
    $Q_{root} \leftarrow$ TREE($T$)
    DTRANN($Q_{root}, R_{root}, \beta$)

DTRANN($Q, R, \beta$)

    **if** $node\_ub(Q) >$
    $dist\_between\_nodes(Q, R)$ **then**
      **if** ISLEAF($Q$) && ISLEAF($R$) **then**
        COMPUTEBRUTENN($Q, R$)
      **else if** ISLEAF($R$) **then**
        DTRANN($Q^l, R, \beta$), DTRANN($Q^r, R, \beta$)
        $node\_ub(Q) \leftarrow \max_{i=\{l,r\}} node\_ub(Q^i)$
      **else if** CANAPPROXIMATE($R, \beta$) **then**
        **if** ISLEAF($Q$) **then**
          COMPUTEAPPROXNN ($Q, R, \beta$)
        **else**
          DTRANN($Q^l, R, \beta$),
          DTRANN($Q^r, R, \beta$)
          $node\_ub(Q) \leftarrow \max_{i=\{l,r\}} node\_ub(Q^i)$
        **end if**
      **else if** ISLEAF($Q$) **then**
        DTRANN($Q, R^l, \beta$), DTRANN($Q, R^r, \beta$)
      **else**
        DTRANN($Q^l, R^l, \beta$), DTRANN($Q^l, R^r, \beta$)
        DTRANN($Q^r, R^l, \beta$),
        DTRANN($Q^r, R^r, \beta$)
        $node\_ub(Q) \leftarrow \max_{i=\{l,r\}} node\_ub(Q^i)$
      **end if**
    **end if**

CANAPPROXIMATE($R, \beta$)
    **return** $\lceil \beta \cdot |R| \rceil \leq$ MAXSAMPLES

Figure 2: Single tree (STRANKAPPROXNN) and dual tree (DTRANKAPPROXNN) algorithms and subroutines for RANN search for a query $q$ (or a query set $T$) in a dataset $S$ with rank approximation $\tau$ and success probability $\alpha$. $R^l$ and $R^r$ are the closer and farther child respectively of $R$ from the query $q$ (or a query node $Q$)

Different datasets drawn for the UCI repository (Bio dataset 300k×74, Corel dataset 40k×32, Covertype dataset 600k×55, Phy dataset 150k×78)[21], MNIST handwritten digit recognition dataset (60k×784)[22] and the Isomap "images" dataset (700×4096)[3] are used. The final dataset "urand" is a synthetic dataset of points uniform randomly sampled from a unit ball (1m×20). This dataset is used to show that even in the absence of a lower-dimensional subspace, RANN is able to get significant speedups over exact methods for relatively low errors. For each dataset, the NN of every point in the dataset is found in the exact case, and $(1 + \lceil \varepsilon \cdot N \rceil)$-rank-approximate NN of every point in the dataset is found in the approximate case. These results are summarized in Fig.3.

The results show that for even low values of $\varepsilon$ (high accuracy setting), the RANN algorithm is significantly more scalable than the exact algorithms for all the datasets. Note that for some of the datasets, the low values of approximation used in the experiments are equivalent to zero rank error (which is the exact case), hence are equally efficient as the exact algorithm.

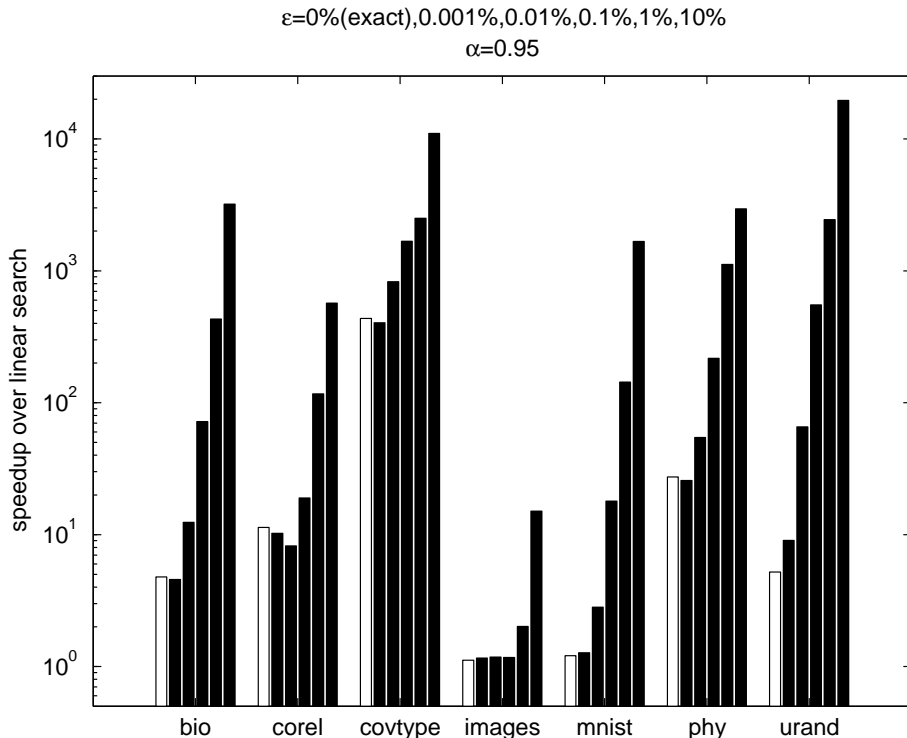

ε=0%(exact),0.001%,0.01%,0.1%,1%,10%
α=0.95

Figure 3: Speedups(logscale on the Y-axis) over the linear search algorithm while finding the NN in the exact case or $(1 + \varepsilon N)$-RANN in the approximate case with $\varepsilon = 0.001\%, 0.01\%, 0.1\%, 1.0\%, 10.0\%$ and a fixed success probability $\alpha = 0.95$ for every point in the dataset. The first(white) bar in each dataset in the X-axis is the speedup of exact dual tree NN algorithm, and the subsequent(dark) bars are the speedups of the approximate algorithm with increasing approximation.

## 4.2 Comparison with Distance-Approximate Search

In the case of the different forms of approximation, the average rank errors and the maximum rank errors achieved in comparable retrieval times are considered for comparison. The rank errors are compared since any method with relatively lower rank error will obviously have relatively lower distance error. For DANN, Locality Sensitive Hashing (LSH) [19, 18] is used.

Subsets of two datasets known to have a lower-dimensional embedding are used for this experiment - Layout Histogram (10k×30)[21] and MNIST dataset (10k×784)[22]. The approximate NN of every point in the dataset is found with different levels of approximation for both the algorithms. The average rank error and maximum rank error is computed for each of the approximation levels. For our algorithm, we increased the rank error and observed a corresponding decrease in the retrieval time. LSH has three parameters. To obtain the best retrieval times with low rank error, we fixed one parameter and changed the other two to obtain a decrease in runtime and did this for many values of the first parameter. The results are summarized in Fig. 4 and Fig. 5.

The results show that even in the presence of a lower-dimensional embedding of the data, the rank errors for a given retrieval time are comparable in both the approximate algorithms. The advantage of the rank-approximate algorithm is that the rank error can be directly controlled, whereas in LSH, tweaking in the cross-product of its three parameters is typically required to obtain the best ranks for a particular retrieval time. Another advantage of the tree-based algorithm for RANN is the fact that even though the maximum error is bounded only with a probability, the actual maximum error is not much worse than the allowed maximum rank error since a tree is used. In the case of LSH, at times, the actual maximum rank error is extremely large, corresponding to LSH returning points which are very far from being the NN. This makes the proposed algorithm for RANN much more stable

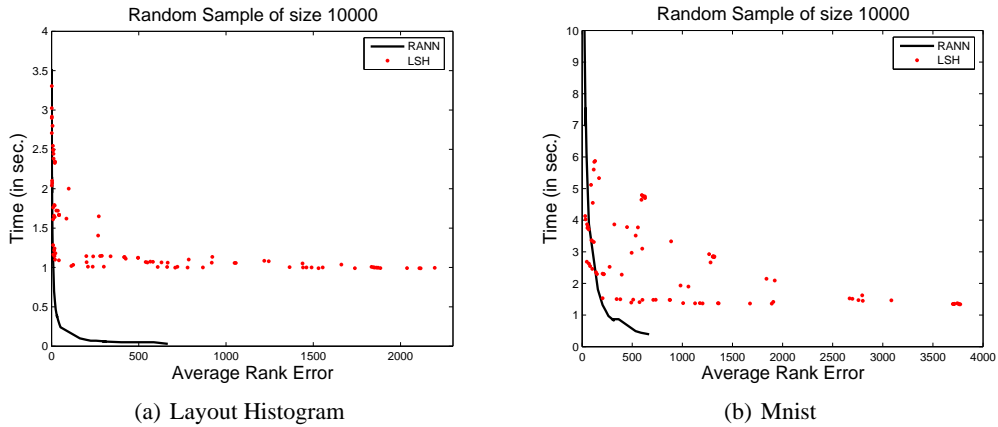

(a) Layout Histogram          (b) Mnist

Figure 4: Query times on the X-axis and the Average Rank Error on the Y-axis.

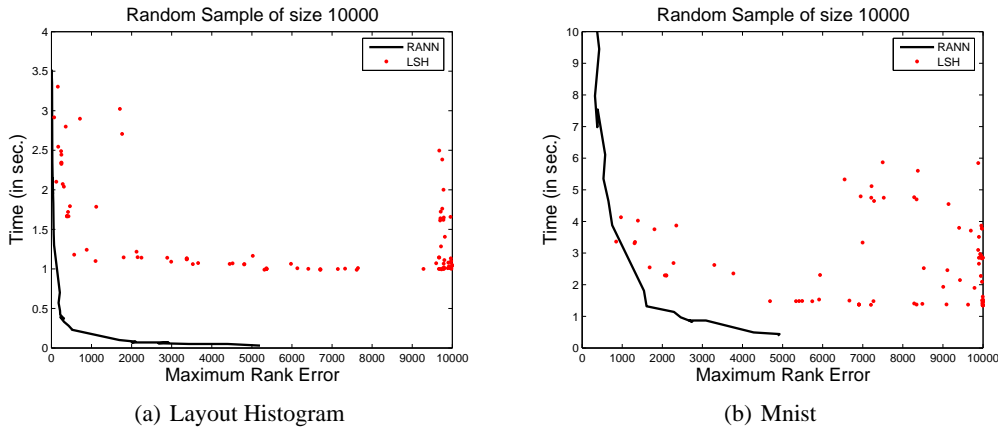

(a) Layout Histogram          (b) Mnist

Figure 5: Query times on the X-axis and the Maximum Rank Error on the Y-axis.

than LSH for Euclidean NN search. Of course, the reported times highly depend on implementation details and optimization tricks, and should be considered carefully.

## 5 Conclusion

We have proposed a new form of approximate algorithm for unscalable NN search instances by controlling the true error of NN search (i.e. the ranks). This allows approximate NN search to retain meaning in high dimensional datasets even in the absence of a lower-dimensional embedding. The proposed algorithm for approximate Euclidean NN has been shown to scale much better than the exact algorithm even for low levels of approximation even when the true dimension of the data is relatively high. When compared with the popular DANN method (LSH), it is shown to be comparably efficient in terms of the average rank error even in the presence of a lower dimensional subspace of the data (a fact which is crucial for the performance of the distance-approximate method). Moreover, the use of spatial-partitioning tree in the algorithm provides stability to the method by clamping the actual maximum error to be within a reasonable rank threshold unlike the distance-approximate method.

However, note that the proposed algorithm still benefits from the ability of the underlying tree data structure to bound distances. Therefore, our method is still not necessarily immune to the curse of dimensionality. Regardless, RANN provides a new paradigm for NN search which is comparably efficient to the existing methods of distance-approximation and allows the user to directly control the true accuracy which is present in ordering of the neighbors.

# References

[1] T. Hastie, R. Tibshirani, and J. H. Friedman. *The Elements of Statistical Learning: Data Mining, Inference, and Prediction*. Springer, 2001.

[2] B. W. Silverman. *Density Estimation for Statistics and Data Analysis*. Chapman & Hall/CRC, 1986.

[3] J. B. Tenenbaum, V. Silva, and J.C. Langford. A Global Geometric Framework for Nonlinear Dimensionality Reduction. *Science*, 290(5500):2319–2323, 2000.

[4] S. T. Roweis and L. K. Saul. Nonlinear Dimensionality Reduction by Locally Linear Embedding. *Science*, 290(5500):2323–2326, December 2000.

[5] A. N. Papadopoulos and Y. Manolopoulos. *Nearest Neighbor Search: A Database Perspective*. Springer, 2005.

[6] N. Alon, M. Bǎdoiu, E. D. Demaine, M. Farach-Colton, and M. T. Hajiaghayi. Ordinal Embeddings of Minimum Relaxation: General Properties, Trees, and Ultrametrics. 2008.

[7] K. Beyer, J. Goldstein, R. Ramakrishnan, and U. Shaft. When Is "Nearest Neighbor" Meaningful? *LECTURE NOTES IN COMPUTER SCIENCE*, pages 217–235, 1999.

[8] J. M. Hammersley. The Distribution of Distance in a Hypersphere. *Annals of Mathematical Statistics*, 21:447–452, 1950.

[9] J. H. Freidman, J. L. Bentley, and R. A. Finkel. An Algorithm for Finding Best Matches in Logarithmic Expected Time. *ACM Trans. Math. Softw.*, 3(3):209–226, September 1977.

[10] S. M. Omohundro. Five Balltree Construction Algorithms. Technical Report TR-89-063, International Computer Science Institute, December 1989.

[11] F. P. Preparata and M. I. Shamos. *Computational Geometry: An Introduction*. Springer, 1985.

[12] A. Beygelzimer, S. Kakade, and J.C. Langford. Cover Trees for Nearest Neighbor. *Proceedings of the 23rd international conference on Machine learning*, pages 97–104, 2006.

[13] A. G. Gray and A. W. Moore. '$N$-Body' Problems in Statistical Learning. In *NIPS*, volume 4, pages 521–527, 2000.

[14] T. Liu, A. W. Moore, A. G. Gray, and K. Yang. An Investigation of Practical Approximate Nearest Neighbor Algorithms. In *Advances in Neural Information Processing Systems 17*, pages 825–832, 2005.

[15] L. Cayton. Fast Nearest Neighbor Retrieval for Bregman Divergences. *Proceedings of the 25th international conference on Machine learning*, pages 112–119, 2008.

[16] T. Liu, A. W. Moore, and A. G. Gray. Efficient Exact k-NN and Nonparametric Classification in High Dimensions. 2004.

[17] P. Ciaccia and M. Patella. PAC Nearest Neighbor Queries: Approximate and Controlled Search in High-dimensional and Metric spaces. *Data Engineering, 2000. Proceedings. 16th International Conference on*, pages 244–255, 2000.

[18] A. Gionis, P. Indyk, and R. Motwani. Similarity Search in High Dimensions via Hashing. pages 518–529, 1999.

[19] P. Indyk and R. Motwani. Approximate Nearest Neighbors: Towards Removing the Curse of Dimensionality. In *STOC*, pages 604–613, 1998.

[20] J. Sedransk and J. Meyer. Confidence Intervals for the Quantiles of a Finite Population: Simple Random and Stratified Simple Random sampling. *Journal of the Royal Statistical Society*, pages 239–252, 1978.

[21] C. L. Blake and C. J. Merz. UCI Machine Learning Repository. http://archive.ics.uci.edu/ml/, 1998.

[22] Y. LeCun. MNIST dataset, 2000. http://yann.lecun.com/exdb/mnist/.

